# A Computational Mechanism To Account For Averaged Modified Hand Trajectories

**Ealan A. Henis*and Tamar Flash**
Department of Applied Mathematics and Computer Science
The Weizmann Institute of Science
Rehovot 76100, Israel

## Abstract

Using the double-step target displacement paradigm the mechanisms underlying arm trajectory modification were investigated. Using short (10-110 msec) inter-stimulus intervals the resulting hand motions were initially directed in between the first and second target locations. The kinematic features of the modified motions were accounted for by the superposition scheme, which involves the vectorial addition of two independent point-to-point motion units: one for moving the hand toward an internally specified location and a second one for moving between that location and the final target location. The similarity between the inferred internally specified locations and previously reported measured end-points of the first saccades in double-step eye-movement studies may suggest similarities between perceived target locations in eye and hand motor control.

## 1  INTRODUCTION

The generation of reaching movements toward unexpectedly displaced targets involves more complicated planning and control problems than in reaching toward stationary ones, since the planning of the trajectory modification must be performed before the initial plan is entirely completed. One possible scheme to modify a trajectory plan is to abort the rest of the original motion plan, and replace it with a new one for moving toward the new target location. Another possible modifica-

tion scheme is to superimpose a second plan with the initial one, without aborting it. Both schemes are discussed below.

Earlier studies of reaching movements toward static targets have shown that point-to-point reaching hand motions follow a roughly straight path, having a typical bell-shaped velocity profile. The kinematic features of these movements were successfully accounted for (Figure 1, left) by the minimum-jerk model (Flash & Hogan, 1985). In that model the $X$-components of hand motions (and analogously the $Y$-components) were represented by:

$$X(t) = X_A + (X_B - X_A)(10T^3 - 15T^4 + 6T^5), \quad T = \frac{t}{t_f} \qquad (1)$$

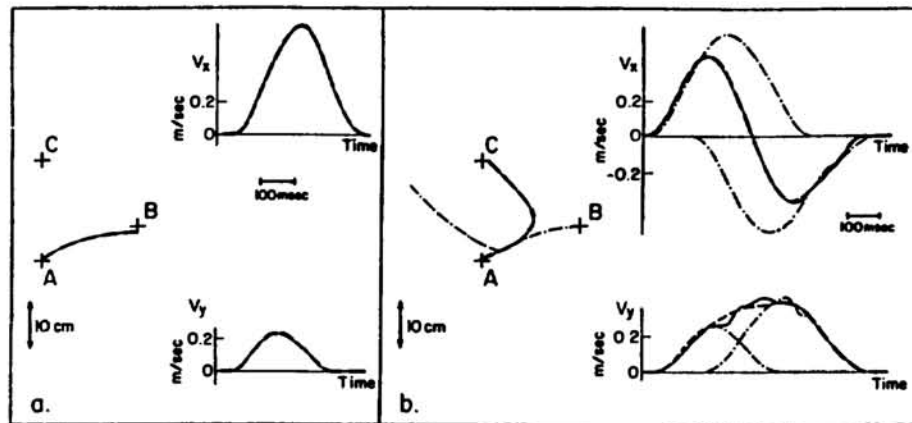

Figure 1: The Minimum-jerk Model and The Non-averaged Superposition Scheme

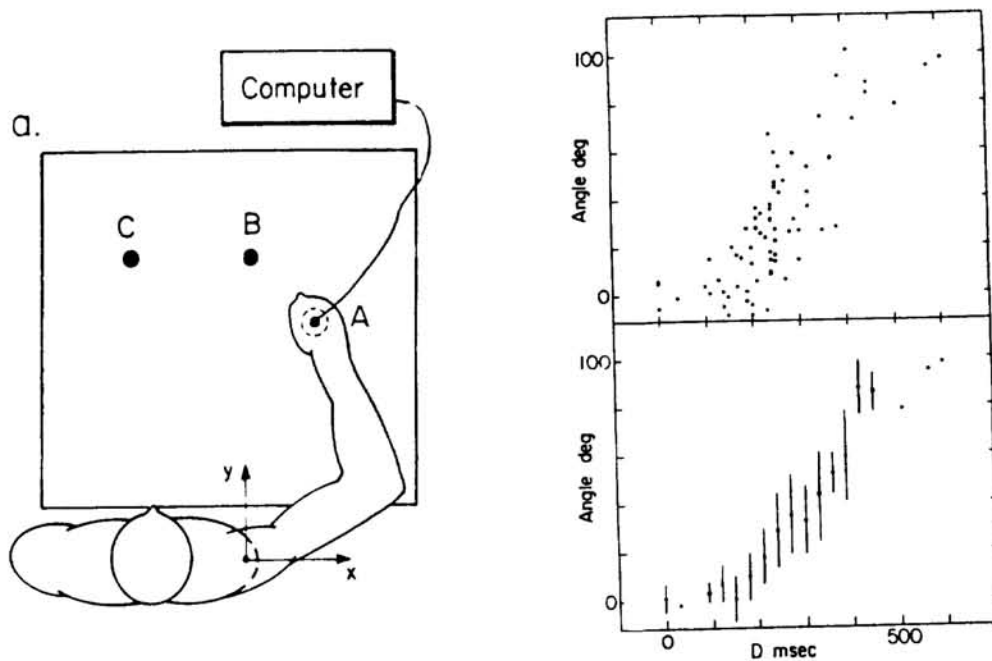

Figure 2: The Experimental Setup and The Initial Movement Direction Vs. D

where $t_f$ is the movement duration, and $X_B - X_A$ is the $X$-component of movement amplitude. In a previous study (Henis & Flash, 1989; Flash & Henis, 1991) we have used the double-step target displacement paradigm (see below) with inter-stimulus intervals (ISIs) of 50-400 msec. Many of the resulting motions were found to be initially directed toward the first target location (*non-averaged*) (for larger ISIs a larger percentage of the motions were non-averaged). The kinematic features of these modified motions were successfully accounted for (Figure 1 right) by a superposition modification scheme that involves the vectorial addition of two time-shifted independent point-to-point motion units (Equation (1)) that have amplitudes that correspond to the two target displacements.

In the present study shorter ISIs of 10-110 msec were used, hence, all target displacements occurred before movement initiation. Most of the resulting hand motions were found to be initially directed in between the first and second target locations (*averaged* motions). For increasing values of $D$, where $D = RT_1 - ISI$ ($RT_1$ is the reaction time), the initial motion direction gradually shifted from the direction of the first toward the direction of the second stimulus (Figure 2 right). The averaging phenomenon has been previously reported for hand (Van Sonderen et al., 1988) and eye (Aslin & Shea, 1987; Van Gisbergen et al., 1987) motions. In this work we wished to account for the kinematic features of averaged trajectories as well as for the dependence of their initial direction on $D$.

It was observed (Van Sonderen et al., 1988) that when the first target displacement was toward the left and the second one was obliquely downwards and to the right most of the resulting motions were *averaged*. Averaged motions were also induced when the first target displacement was downwards and the second one was obliquely upwards and to the left. In this study we have used similar target displacements. Four naive subjects participated in the experiments. The motions were performed in the absence of visual feedback from the moving limb. In a typical trial, initially the hand was at rest at a starting position $A$ (Figure 2 left). At $t = 0$ a visual target was presented at one of two equally probable positions $B$. It either remained lit (control condition, probability 0.4) or was shifted again, following an ISI, to one of two equally probable positions $C$ (double-step condition, probability 0.3 each). In a block of trials one target configuration was used. Each block consisted of five groups of 56 trials, and within each group one ISI pair was used. The five ISI pairs were: 10 and 80, 20 and 110, 30 and 150, 40 and 200, and 50 and 300 msec. The target presentation sequence was randomized, and included appropriate control trials.

## 2    MODELING RATIONALE AND ANALYSIS

### 2.1    THE SUPERPOSITION SCHEME

The superposition scheme for *averaged* modified motions is based on the vectorial addition of two time-shifted independent elemental point-to-point hand motions that obey Equation (1). The first elemental trajectory plan is for moving between the initial hand location and an intermediate location $B_i$, internally specified. This plan continues unmodified until its intended completion. The second elemental trajectory plan is for moving between $B_i$ and the final location of the target. The durations of the elemental motions may vary among trials, and are therefore a

priori unknown. With short ISIs the elemental motion plans may be added (to give the modified plan) preceding movement initiation. Several possibilities for $B_i$ were examined: a) the first location of the stimulus, b) an a priori unknown position, c) same as (b) with $B_i$ constrained to lie on the line connecting the first and second locations of the stimulus, and d) same as (b) with $B_i$ constrained to lie on the line of initial movement direction. Version (a) is equivalent to the superposition scheme that successfully accounted for *non-averaged* modified trajectories (Flash & Henis, 1991). In versions (b), (c) and (d) it was assumed that due to the quick displacement of the target, the specification of the end-point for the first motion plan may differ from the actual first location of the target. The first motion unit was represented by:

$$X_1(t) = X_A + (X_{B_i} - X_A)(10T^3 - 15T^4 + 6T^5), \quad \text{where} \quad T = \frac{t}{T_1}. \quad (2)$$

In (2), $(X_{B_i} - X_A)$ is the $X$-component of the first unit amplitude. The duration of this unit is denoted by $T_1$, a priori unknown. The expression for $Y_1(t)$ was analogous to Equation (2). The $X$-component of the second motion unit was taken to be:

$$X_2(t) = (X_C - X_{B_i})(10T^3 - 15T^4 + 6T^5), \quad \text{where} \quad T = \frac{t - t_s}{t_f - t_s} = \frac{t - t_s}{T_2}. \quad (3)$$

In (3), $(X_C - X_{B_i})$ is the $X$-component of the amplitude of the second trajectory unit. The start and end times of the second unit are denoted by $t_s$ and $t_f$, respectively. The duration of the second motion unit $T_2 = t_f - t_s$ is a priori unknown. The $X$-component of the entire modified motion (and similarly for the $Y$-component) was represented by:

$$X(t) = X_1(t) + X_2(t). \quad (4)$$

The unknown parameters $T_1, T_2, B_{iX}$ and $B_{iY}$ that can best describe the entire measured trajectory were determined by using least-squares best-fit methods (Marquardt, 1963). This procedure minimized the sum of the position errors between the simulated and measured data points, taking into account (in versions (a), (c) and (d)) the assumed constraints on the location $B_i$.

## 2.2   THE ABORT-REPLAN SCHEME

In the abort-replan scheme it was assumed that initially a point-to-point trajectory plan is generated for moving toward an initial target (Equation (2)). The same four different possibilities for the end-point of the initial motion plan were examined. It was assumed that at some time-instant $t_s$ the initial plan is aborted and replaced by a new plan for moving between the expected hand position at $t = t_s$ and the final target location. The new motion plan was assumed to be represented by:

$$X_{NEW}(t) = \sum_{i=0}^{5} a_i(t)^i. \quad (5)$$

The coefficients $a_3, a_4$ and $a_5$ were derived by using the the measured values of position, velocity and acceleration at $t = t_f$. For versions (b), (c) and (d) the analysis was performed simultaneously for the $X$ and $Y$ components of the trajectory. Choosing a trial $B_i$ and $T_1$ the initial motion plan (Equation (2)) was

calculated. Choosing a trial $t_s$, the remaining three unknown coefficients $a_0, a_1$ and $a_2$ of Equation (5) were calculated using the continuity conditions of the initial and new position, velocity and acceleration at $t = t_s$ (method I). Alternatively, these three coefficients were calculated using the corresponding *measured* values at $t = t_s$ (method II). To determine the best choice of the unknown parameters $B_{iX}, B_{iY}, T_1$ and $t_s$ the same least squares methods (Marquardt, 1963) were used as described above. For version (a), for each cartesian component, a point-to-point minimum-jerk trajectory $AB$ was speed-scaled to coincide with the initial part of the measured velocity profile. The time $t_s$ of its deviation from the measured speed profile was extracted. From $t_s$ on, the trajectory was represented by Equation (5). The values of $a_0, a_1$ and $a_2$ were derived by using the same least squares methods (method I). Alternatively, these values were determined by using the measured position, velocity and acceleration at $t = t_s$ (method II).

## 3    RESULTS

The motions recorded in the control trials were roughly straight with bell-shaped speed profiles. The mean reaction time in these control trials was $367.1 \pm 94.6$ msec ($N = 120$). The mean movement time was $574.1 \pm 127.0$ msec ($N = 120$). The change in target location elicited a graded movement toward an intermediate direction in between the two target locations, followed by a subsequent motion toward the final target (Figure 3, middle). Occasionally the hand went directly toward the final target location (Figure 3, right). For values of $D$ less than 100 ms the movements were found to be initially directed toward the first target (Figure 3, left). As $D$ increased, the initial direction of the motions gradually shifted (Figure 2, right) from the direction of the first (non-averaged) toward the direction of the second (direct) target locations (The initial direction depended on $D$ rather than on ISI). The mean reaction time to the first stimulus ($RT_1$) was $350.4 \pm 93.5$ msec (N=192). The mean reaction time to the second stimulus ($RT_2$) (inferred from the superposition version (b)) was $382.8 \pm 119.9$ msec (N=192). This value is much smaller than that predicted by successive processing of information: $RT_2 = 2RT_1$-ISI (Poulton, 1981), and might be indicative of the presence of parallel planning. The mean durations $T_1$ and $T_2$ of the two trajectory units (of superposition version (b)) were: $373.0 \pm 112.2$ and $592.1 \pm 98.1$ msec ($N = 192$), respectively.

### 3.1    MODIFICATION SCHEMES

The most statistically successful model (Table-1) in accounting for the measured motions was the superposition version (b), which involves an internally specified location (a priori unknown) for the end-point of the first motion unit. In particular, the averaged initial direction of the measured motions was accounted for. Superposition version (d) was equivalent to version (b). The velocities simulated on the basis the other tested schemes substantially deviated from the measured ones (Table 1 and Figure 4). It should be noted that in both the superposition and abort-replan versions (b), (c) and (d) there were 4, 3 and 3 unknown parameters. In the abort-replan versions (aI) there were 3 unknown parameters, compared to 2 in the superposition version (a). Hence the relative success of the superposition version (b) in accounting for the data was not due to a larger number of free parameters.

Table 1: Normalized Velocity Deviations and The t-score With SP(b))

| SP(a) | SP(b) | SP(c) | SP(d) | AB(aI) | AB(aII) | AB(bI) | AB(bII) | AB(cI) | AB(cII) | AB(dI) | AB(dII) |
|---|---|---|---|---|---|---|---|---|---|---|---|
| 18.60 | 0.035 | 0.126 | 0.042 | 0.083 | 0.084 | 0.081 | 0.078 | 0.084 | 0.083 | 0.082 | 0.085 |
| ± 50.16 | ± 0.036 | ± 0.132 | ± 0.045 | ± 0.093 | ± 0.088 | ± 0.101 | ± 0.102 | ± 0.108 | ± 0.096 | ± 0.097 | ± 0.101 |
| (4.711) | (0.000) | (8.465) | (1.546) | (6.126) | (6.559) | (5.460) | (5.050) | (5.478) | (5.959) | (5.782) | (5.935) |

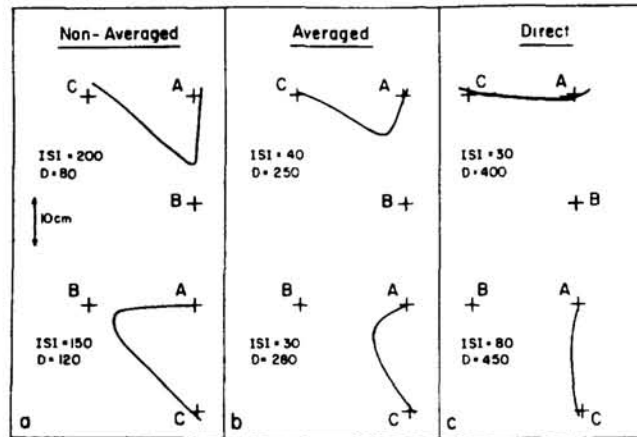

Figure 3: Types of Modified Trajectories

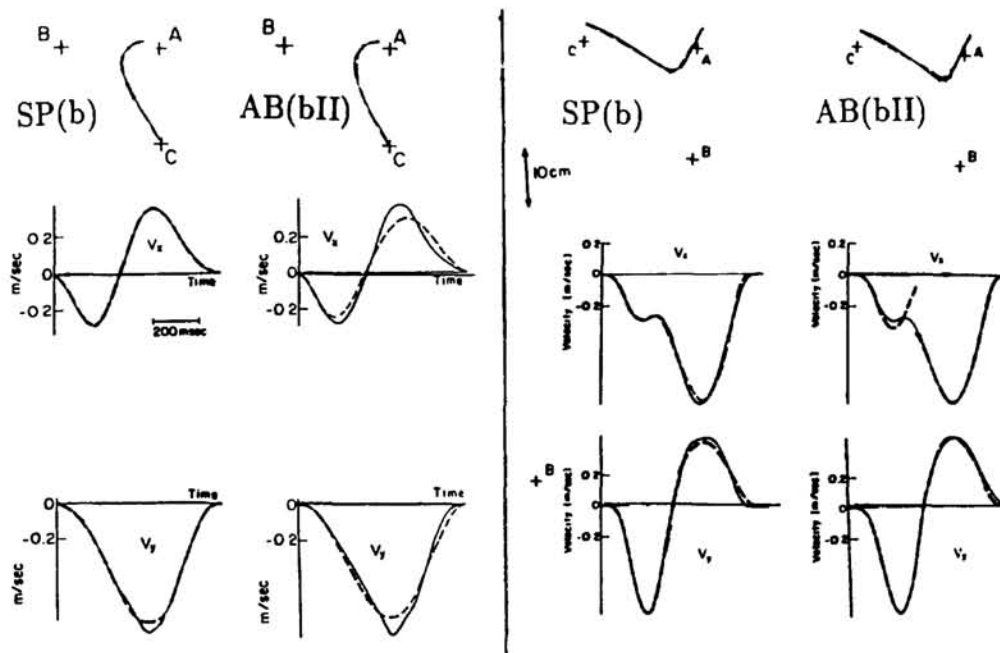

Figure 4: Representative Simulated Vs. Measured Trajectories

## 3.2  THE END-POINTS INFERRED FROM SUPERPOSITION (b)

The mean locations $B_i$ resulting from different trials performed by the same subject were computed by pooling together $B_i$ of movements with the same $D \pm 15$ msec (Figure 5 left). For $D \leq 100$ msec, the measured motions were non-averaged and the inferred $B_i$ were in the vicinity of the first target. For increasing values of $D$, $B_i$ gradually shifted from the first toward the second target location, following a typical path that curved toward the initial hand position. For $D \geq 400$ msec, $B_i$ were in the vicinity of the second target location. Since initially the motions are directed toward $B_i$, this gradual shift of $B_i$ as a function of $D$ may account for the observed dependence of the initial direction of motion on $D$. The locations $B_i$ obtained on the basis of the other tested schemes did not show any regular behavior as functions of $D$.

## 4  DISCUSSION

This paper presents explicit possible mechanisms to account for the kinematic features of averaged modified trajectories. the most statistically successful scheme in accounting for the measured movements involves the vectorial addition of two independent point-to-point motion units: one for moving between the initial hand position and an internally specified location, and a second one for moving between that location and the final target location. Taken together with previous results for *non-averaged* modified trajectories (Flash & Henis, 1991), it was shown that the same superposition principle may account for both modified trajectory types. The differences between the observed types stem from differences in the time available to modify the end-point of the first unit. Our simulations have enabled us to infer the locations of the intermediate target locations, which were found to be similar to previously reported (Aslin & Shea, 1987) experimentally measured end-points of the first saccades, obtained by using the double-step paradigm (Figure 5 right[1]). This result may suggests underlying similarities between internally perceived target locations in eye and hand motor control and may suggest a common "where" command (Gielen et al., 1984; 1990) for both systems.

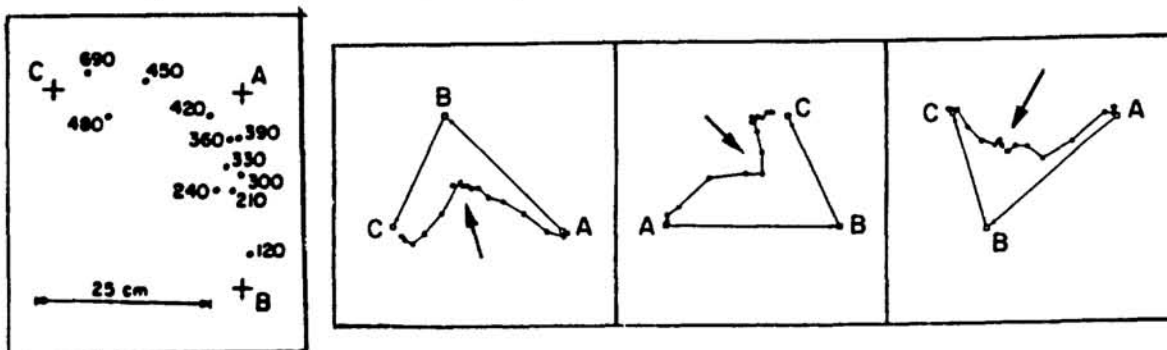

Figure 5: Inferred First Unit End-points and Measured Eye Positions

Why is the internally specified location dependent on $D$, which is a parameter associated with both sensory information and motor execution? One possible explanation is that following the target displacement the effect of the first stimulus on the motion planning decays, and that of the second stimulus becomes larger. These changes may occur in the transformations from the visual to the motor system. A purely sensory change in the perceived target location was also proposed (Van Sonderen et al., 1988; Becker & Jurgens 1979). Another possibility is that the direction of hand motion is internally coded in the motor system (Georgopoulos et al., 1986), and it gradually rotates (within the motor system) from the direction of the first to the direction of the second target. It is not clear which of these possibilities provides a better explanation for the observations.

In the superposition scheme there is no need to keep track of the actual or planned kinematic state of the hand. Hence, in contrast to the abort-replan scheme, an efference copy of the planned motion is not required. The successful use of motion plans expressed in extrapersonal coordinates provides support to the idea that arm movements are internally represented in terms of hand motion through external space. The construction of complex movements from simpler elementary building blocks is consistent with a hierarchical organization of the motor system. The independence of the elemental trajectories allows to plan them in parallel.

## Acknowledgements

This research was supported by a grant no. 8800141 from the United-States Israel Binational Science Foundation (BSF), Jerusalem, Israel. Tamar Flash is incumbent of the Corinne S. Koshland career development chair.

## Footnotes

*Current address IRCS/GRASP, University of Pennsylvania.

[1]Reprinted with permission from *Vision Res., Vol. 27, No. 11*, 1925-1942, Aslin, R.N. and Shea S.L.: The Amplitude And Angle of Saccades to Double-Step Target Displacements, Copyright [1987], Pergamon Press plc.

## References

Aslin, R.N. and Shea S.L. (1987). The Amplitude And Angle of Saccades to Double-Step Target Displacements. Vision Res., Vol. 27, No. 11, 1925-1942.

Becker W. and Jurgens R. (1979). An Analysis of The Saccadic System By Means of Double-Step Stimuli. Vision Res., 19, 967-983.

Flash T. and Henis E. (1991). Arm Trajectory Modification During Reaching Towards Visual Targets. Journal of Cognitive Neuroscience Vol. 3, no. 3, 220-230.

Flash, T. & Hogan, N. (1985). The coordination of arm movements: an experimentally confirmed mathematical model. J. Neurosci., 7, 1688-1703.

Georgopoulos A.P., Schwartz A.B. & Kettner R.E. (1986). Neuronal population coding of movement direction. Science 233, 1416-1419.

Gielen, C.C.A.M., Van den Heuvel, P.J.M. & Denier Van der Gon, J.J. (1984). Modification of muscle activation patterns during fast goal-directed arm movements. J. Motor Behavior, 16, 2-19.

Gielen C.C.A.M. & Van Gisbergen J.A.M. (1990). The visual guidance of saccades and fast aiming movements. News in Physiological Sciences Vol.5, 58-63.

Henis E. and Flash T. (1989). Mechanisms Subserving Arm Trajectory Modification. Perception 18(4):495.

Marquardt, D.W., (1963). An algorithm for least-squares estimation of non-linear parameters. J. SIAM, 11, 431-441.

Van Gisbergen, J.A.M., Van Opstal, A.J. & Roebroek, J.G.H. (1987). Stimulus-induced midflight modification of saccade trajectories. In J.K. O'Regan & A. Levy-Schoen (Eds.), Eye Movements: From Physiology to Cognition, Amsterdam: Elsevier, 27-36.

Van Sonderen, J.F., Denier Van Der Gon, J.J. & Gielen, C.C.A.M. (1988). Conditions determining early modification of motor programmes in response to change in target location. Exp. Brain Res., 71, 320-328.